# Using Aperiodic Reinforcement for Directed Self-Organization During Development

PR Montague   P Dayan   SJ Nowlan   A Pouget   TJ Sejnowski
CNL, The Salk Institute
10010 North Torrey Pines Rd.
La Jolla, CA 92037, USA
read@helmholtz.sdsc.edu

## Abstract

We present a local learning rule in which Hebbian learning is conditional on an incorrect prediction of a reinforcement signal. We propose a biological interpretation of such a framework and display its utility through examples in which the reinforcement signal is cast as the delivery of a neuromodulator to its target. Three examples are presented which illustrate how this framework can be applied to the development of the oculomotor system.

## 1   INTRODUCTION

Activity-dependent accounts of the self-organization of the vertebrate brain have relied ubiquitously on correlational (mainly Hebbian) rules to drive synaptic learning. In the brain, a major problem for any such unsupervised rule is that many different kinds of correlations exist at approximately the same time scales and each is effectively noise to the next. For example, relationships within and between the retinae among variables such as color, motion, and topography may mask one another and disrupt their appropriate segregation at the level of the thalamus or cortex.

It is known, however, that many of these variables can be segregated both within and between cortical areas suggesting that certain sets of correlated inputs are somehow separated from the temporal noise of other inputs. Some form of supervised learning appears to be required. Unfortunately, **detailed supervision and**

**selection** in a brain region is not a feasible mechanism for the vertebrate brain. The question thus arises: What kind of biological mechanism or signal could selectively bias synaptic learning toward a particular subset of correlations ? One answer lies in the possible role played by diffuse neuromodulatory systems.

It is known that multiple diffuse modulatory systems are involved in the self-organization of cortical structures (*eg* Bear and Singer, 1986) and some of them appear to deliver reward and/or salience signals to the cortex and other structures to influence learning in the adult. Recent data (Ljunberg, *et al*, 1992) suggest that this latter influence is qualitatively similar to that predicted by Sutton and Barto's (1981,1987) classical conditioning theory. These systems innervate large expanses of cortical and subcortical turf through extensive axonal projections that originate in midbrain and basal forebrain nuclei and deliver such compounds as dopamine, serotonin, norepinephrine, and acetylcholine to their targets. The small number of neurons comprising these subcortical nuclei relative to the extent of the territory their axons innervate suggests that the nuclei are reporting scalar signals to their target structures.

In this paper, these facts are synthesized into a single framework which relates the development of brain structures and conditioning in adult brains. We postulate a modification to Hebbian accounts of self-organization: Hebbian learning is conditional on a incorrect prediction of future delivered reinforcement from a diffuse neuromodulatory system. This reinforcement signal can be derived both from externally driven contingencies such as proprioception from eye movements as well as from internal pathways leading from cortical areas to subcortical nuclei.

The next section presents our framework and proposes a specific model for how predictions about future reinforcement could be made in the vertebrate brain utilizing the firing in a diffuse neuromodulatory system (figure 1). Using this model we illustrate the framework with three examples suggesting how mappings in the oculomotor system may develop. The first example shows how eye movement commands could become appropriately calibrated in the absence of visual experience (figure 3). The second example demonstrates the development of a mapping from a selected visual target to an eye movement which acquires the target. The third example describes how our framework could permit the development and alignment of multimodal maps (visual and auditory) in the superior colliculus. In this example, the transformation of auditory signals from head-centered to eye-centered coordinates results implicitly from the development of the mapping from parietal cortex onto the colliculus.

## 2   THEORY

We consider two classes of reinforcement learning (RL) rule: static and dynamic.

### 2.1   Static reinforcement learning

The simplest learning rule that incorporates a reinforcement signal is:

$$\Delta w_t = \alpha x_t y_t r_t \tag{1}$$

where, all at times t, $w_t$ is a connection weight, $x_t$ an input measure, $y_t$ an output measure, $r_t$ a reinforcement measure, and $\alpha$ is the learning rate.

In this case, $r$ can be driven by either external events in the world or by cortical projections (internal events) and it picks out those correlations between $x$ and $y$ about which the system learns. Learning is shut down if nothing occurs that is independently judged to be significant, *i.e.* events for which r is 0.

## 2.2 Dynamic Reinforcement learning - learning driven by prediction error

A more informative way to utilize reinforcement signals is to incorporate some form of prediction. The predictive form of RL, called temporal difference learning (TD, Sutton and Barto, 1981,1987), specifies weight changes according to:

$$\Delta w_t = \alpha x_t[(r_{t+1} + V_{t+1}) - V_t] \tag{2}$$

where $r_{t+1}$ is the reward delivered in the next instant in time $t + 1$. V is called a value function and its value at any time t is an estimate of the future reward. This framework is closely related to dynamic programming (Barto *et al*, 1989) and a body of theory has been built around it. The prediction error $[(r_{t+1} + V_{t+1}) - V_t]$, measures the degree to which the prediction of future reward $V_t$ is higher or lower than the combination of the actual future reward $r_{t+1}$ and the expectation of reward from time $t + 1$ onward ($V_{t+1}$).

To place dynamic RL in a biological context, we start with a simple Hebbian rule but make learning contingent on this prediction error. Learning therefore slows as the predictions about future rewards get better. In contrast with static RL, in a TD account the value of r *per se* is not important, only whether the system is able to predict or anticipate the the future value of r. Therefore the weight changes are:

$$\Delta w_t = \alpha x_t y_t[(r_{t+1} + V_{t+1}) - V_t] \tag{3}$$

including a measure of post-synaptic response, $y_t$.

# 3   MAKING PREDICTIONS IN THE BRAIN

In our account of RL in the brain, the cortex is the structure that makes predictions of future reinforcement. This reinforcement is envisioned as the output of subcortical nuclei which deliver various neuromodulators to the cortex that permit Hebbian learning. Experiments have shown that various of these nuclei, which have access to cortical representations of complex sensory input, are necessary for instrumental and classical conditioning to occur (Ljunberg *et al.*, 1992).

Figure 1 shows one TD scenario in which a pattern of activity in a region of cortex makes a prediction about future expected reinforcement. At time t, the prediction of future reward $V_t$ is viewed as an excitatory drive from the cortex onto one or more subcortical nuclei (pathway B). The high degree of convergence in B ensures that this drive predicts only a scalar output of the nucleus R. Consider a pattern of activity onto layer II which provides excitatory drive to R and concomitantly causes some output, say a movement, at time $t + 1$. This movement provides a separate source of excitatory drive $r_{t+1}$ to the same nucleus through independent

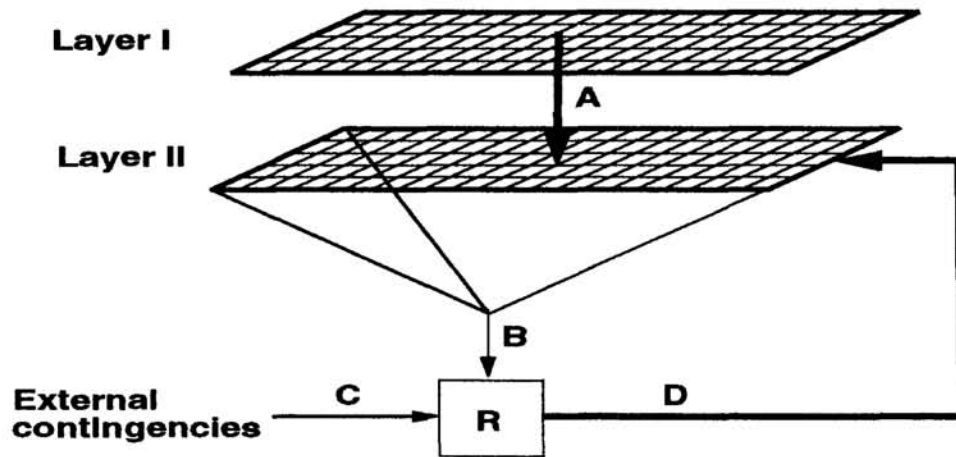

Figure 1: **Making predictions about future reinforcement.** Layer I is an array of units that projects topographically onto layer II. **(A)** Weights from I onto II develop according to equation 3 and represent the value function $V_t$. **(B)** The weights from II onto R are fixed. The prediction of future reward by the weights onto II is a scalar because the highly convergent excitatory drive from II to the reinforcement nucleus **(R)** effectively sums the input. **(C)** External events in the world provide independent excitatory drive to the reinforcement nucleus. **(D)** Scalar signal which results from the output firing of R and is broadcast throughout layer II. This activity delivers to layer II the neuromodulator required for Hebbian learning. The output firing of R is controlled by temporal changes in its excitatory input and habituates to constant or slowly varying input. This makes for learning in layer II according to equation 3 (see text).

---

connections conveying information from sensory structures such as stretch receptors (pathway C). Hence, at time $t + 1$, the excitatory input to R is the sum of the 'immediate reward' $r_{t+1}$ and the new prediction of future reward $V_{t+1}$. If the reinforcement nucleus is driven primarily by changes in its input over some time window, then the difference between the excitatory drive at time t and $t + 1$, *ie* $[(r_{t+1} + V_{t+1}) - V_t]$ is what its output reflects.

The output is distributed throughout a region of cortex (pathway D) and permits Hebbian weight changes at the individual connections which determine the value function $V_t$. The example hinges on two assumptions: 1) Hebbian learning in the cortex is contingent upon delivery of the neuromodulator, and 2) the reinforcement nucleus is sensitive to temporal changes in its input and otherwise habituates to constant or slowly varying input.

Initially, before the system is capable of predicting future delivery of reinforcement correctly, the arrival of $r_{t+1}$ causes a large learning signal because the prediction error $[(r_{t+1} + V_{t+1}) - V_t]$ is large. This error drives weight changes at synaptic connections with correlated pre- and postsynaptic elements until the predictions come to approximate the actual future delivered reinforcement. Once these predictions become accurate, learning abates. At that point, the system has learned about whatever contingencies are currently controlling reinforcement delivery. For the case in which the delivery of reinforcement is not controlled by any predictable contingencies, Hebbian learning can still occur if the fluctuations of the prediction error have a positive mean.

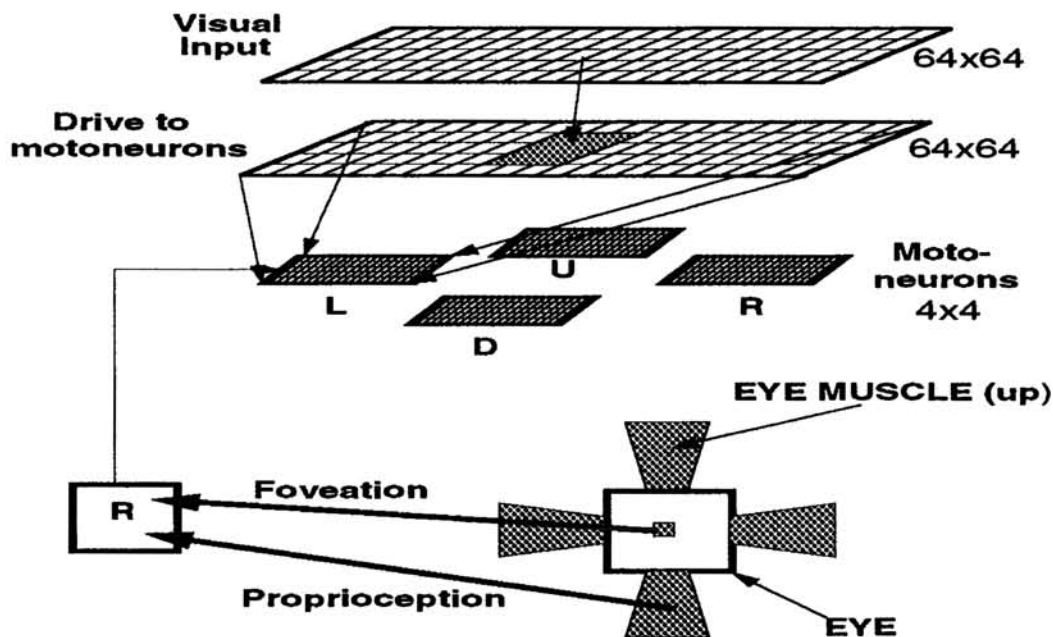

Figure 2: Upper layer is a 64 by 64 input array with 3 by 3 center-surround filters at each position which projects topographically onto the middle layer. The middle layer projects randomly to four 4 X 4 motoneuron layers which code for an equilibrium eye position signal, for example, through setting equilibrium muscle tensions in the 4 muscles. Reinforcement signals originate from either eye movement (muscle 'stretch') or foveation. The eye is moved according to $h = (r - l)g, v = (u - d)g$ where r,l,u,d are respectively the average activities on the right, left, up, down motoneuron layers and g is a fixed gain parameter. h and v are linearly combined to give the eye position.

In the presence of multiple *statistically independent sources of control* of the reinforcement signal (pathways onto R), the system can separately 'learn away' the contingencies for each of these sources. This passage of control of reinforcement delivery can allow the development of connections in a region to be staged. Hence, control of reinforcement can be passed between contingencies without supervision. In this manner, a few nuclei can be used to deliver information globally about many different circumstances. We illustrate this point below with development of a sensorimotor mapping.

## 4   EXAMPLES

### 4.1   Learning to calibrate without sensory experience

Figure 2 illustrates the architecture for the next two examples. Briefly, cortical layers drive four 'motor' layers of units which each provide an equilibrium command to one of four extraocular muscles. The mapping from the cortical layers onto these four layers is random and sparse (15%-35% connectivity) and is plastic according to the learning rule described above. Two external events control the delivery of reinforcement: eye movement and foveation of high contrast objects in the visual input. The minimum eye movement necessary to cause a reinforcement is a change of two pixels in any direction (see figure 3).

We begin by demonstrating how an unbalanced mapping onto the motoneuron

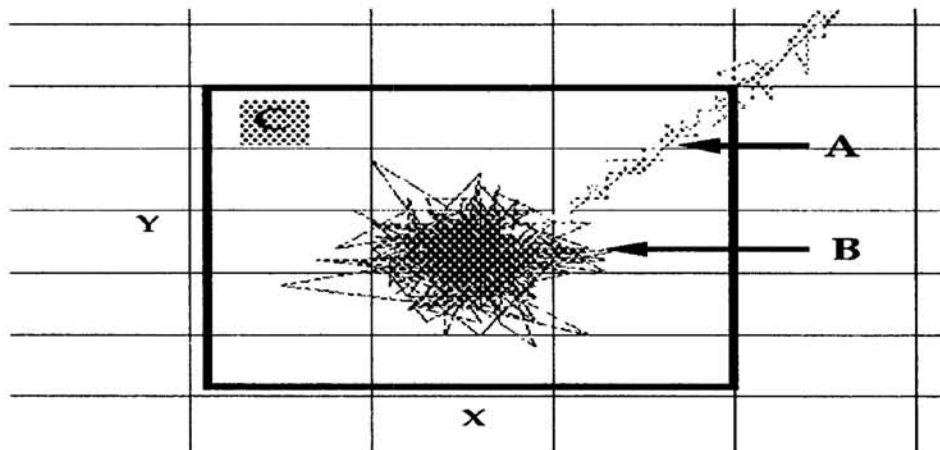

Figure 3: **Learning to calibrate eye movement commands.** This example illustrates how a reinforcement signal could help to organize an appropriate balance in the sensorimotor mapping **before** visual experience. The dark bounding box represents the 64x64 pixel working area over which an 8x8 fovea can move. **A** Foveal position during the first 400 cycles of learning. The architecture is as in figure 2, but the weights onto the right/left and up/down pairs are not balanced. Random activity in the layer providing the drive to the motoneurons initially drives the eye to an extreme position at the upper right. From this position, no movement of the eye can occur and thus no reinforcement can be delivered from the proprioceptive feedback causing all the weights to begin to decrease. With time, the weights onto the motoneurons become balanced and the eye moves. **B** Foveal position after 400 cycles of learning and after increasing the gain g to 10 times its initial value. After the weights onto antagonistic muscles become balanced, the net excursions of the eye are small thus requiring an increase in g in order to allow the eye to explore its working range. **C** Size of foveal region relative the working range of the eye. The fovea covered an 8x8 region of the working area of the eye and the learning rate $\alpha$ was varied from 0.08 to 0.25 without changing the result.

layers can be automatically calibrated in the absence of visual experience. Imagine that the weights onto the right/left and up/down pairs are initially unbalanced, as might happen if one or more muscles are weak or the effective drives to each muscle are unequal. Figure 3, which shows the position of the fovea during learning, indicates that the initially unbalanced weights cause the eye to move immediately to an extreme position (figure 3, A).

Since the reinforcement is controlled only by eye movement and foveation and neither is occurring in this state, $r_{t+1}$ is roughly 0. This is despite the (randomly generated) activity in the motoneurons continually making predictions that reinforcement from eye-movement should be being delivered. Therefore all the weights begin to decrease, with those mediating the unbalanced condition decreasing the fastest, until balance is achieved (see path A). Once the eye reaches equilibrium, further random noise will cause no mean net eye movement since the mappings onto each of the four motoneuron layers are balanced. The larger amplitude eye movements shown in the center of figure 3 (labeled B) are the result of increasing the gain g (figure 2).

Figure 4: **Development of foveation map.** The map after 2000 learning cycles shows the approximate eye movement vector from stimulation of each position in the visual field. Lengths were normalized to the size of the largest movement. The undisplayed quadrants were qualitatively similar. Note that this scheme does not account for activity or contrast differences in the input and assumes that these have already been normalized. Learning rate = 0.12. Connectivity from the middle layer to the motoneurons was 35% and was randomized. Unlike the previous example, the weights onto the four layers of motoneurons were initially balanced.

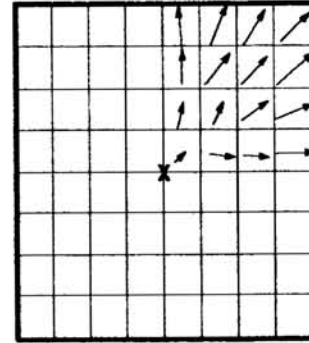

## 4.2  Learning a foveation map with sensory experience

Although reinforcement would be delivered by foveation as well as successful eye-movements, the former would be expected to be a comparatively rare event. Once equilibrium is achieved, however, the reinforcement that comes from eye movements is fully predicted by the prior activity of the motoneurons, and so other contingencies, in this case foveation, grab control of the delivery of reinforcement. The resulting TD signals now provide information about the link between visual input on the top layer of figure 2 and the resulting command, and the system learns how to foveate correctly. Figure 4 shows the motor map that has developed after 2000 learning cycles. In the current example, the weights onto the four layers of motoneurons initially were balanced and the gain g was 10 times larger than before calibration (see figure 3). This learning currently assumes that some cortical area selects the salient targets.

## 4.3  Learning to align separate mappings

In the primate superior colliculus, it is known that cells can respond to multiple modalities including auditory input which defines a head centered coordinate system. Auditory receptive fields shift their position in the colliculus with changing eye position suggesting the existence of a mechanism which maintains the registration between auditory and visual maps (Jay and Sparks, 1984). Our framework suggests a developmental explanation of these findings in terms of an activity-dependent self-organizing principle.

Consider an intermediate layer, modeling the parietal cortex, which receives signals representing eye position (proprioception), retinal position of a visual target (selected visual input), and head position of an auditory target and which projects onto the superior colliculus. This can be visualized using figure 2 with parietal cortex as the top layer and the colliculus as the drive to the motoneurons. As before (figure 2), assume that foveation of a target, whether auditory or visual, delivers reinforcement and that learning in this layer and the colliculus follows equation 3. In a manner analagous to the example in figure 4, those *combinations* of retinal, eye position, and head centered signals in this parietal layer which predict a foveating eye movement are selected by this learning rule. Hence, as before, the weights from this layer onto the colliculus make predictions about future reinforcement. In figure 4, a foveation map develops which codes for eye movements in absolute coordinates relative to some equilibrium position of the eye. In the current example, such a foveation map would be inappropriate since it requires persistent activity in the collicular layer to maintain a fixed eye position. Instead, the collicular to motoneuron mapping must represent changes in the balance between antagonistic muscles with some other system coding for current eye position.

Why would such an initial architecture, acting under the aegis of the learning rule expressed in equation 3, develop the collicular mappings observed in experiments ? Those combinations of signals in the parietal layer that correctly predict foveation have their connections onto the collicular layer stabilized. In the current representation, foveation of a target will occur if the correct *change* in firing between antagonistic motoneurons occurs. After learning slows, the parietal layer is left with cells whose visual and auditory responses are modulated by eye position signals. In the collicular layer, the visual responses of a cell are not modulated by eye position signals while the head-centered auditory responses are modulated by eye position.

The reasons for these differences in the colliculus layer and parietal layer are implicit in the new motoneuron model and the way the equation 3 polices learning. The collicular layer is driven by combinations of the three signals and the learning rule enforces a common frame of reference for these combinations because foveation of the target is the only source of reinforcement. Consider, for example, a visual target on a region of retina for two different eye positions. The change in the balance between right and left muscles required to foveate such a retinal target is the same for each eye position hence the projection from the parietal to collicular layer develops so that the influence of eye position for a fixed retinal target is eliminated. The influence of eye position for an auditory target remains, however, because successful foveation of an auditory target requires different regions of the collicular map to be active as a function of eye position.

These examples illustrate how diffuse modulatory systems in the midbrain and basal forebrain can be employed in single framework to guide activity-dependent map development in the vertebrate brain. This framework gives a natural role to such diffuse system for both development and conditioning in the adult brain and illustrates how external contingencies can be incorporated into cortical representations through these crude scalar signals.

# References

[1] Barto, AG, Sutton, RS & Watkins, CJCH (1989). *Learning and Sequential Decision Making*. Technical Report 89-95, Computer and Information Science, University of Massachusetts, Amherst, MA.

[2] Bear, MF & Singer, W (1986). Modulation of visual cortical plasticity by acetylcholine and noradrenaline. *Nature*, **320**, 172-176.

[3] Jay, MF & Sparks, DL (1984). Auditory receptive fields in primate superior colliculus shift with changes in eye position. *Nature*, **309**, 345-347.

[4] Ljunberg, T, Apicella, P & Schultz, W (1992). Responses of monkey dopamine neurons during learning of behavioral reactions. *Journal of Neurophysiology*, **67**(1), 145-163.

[5] Sutton, RS (1988). Learning to predict by the methods of temporal difference. *Machine Learning*, **3**, pp 9-44.

[6] Sutton, RS & Barto, AG (1981). Toward a modern theory of adaptive networks: Expectation and prediction. *Psychological Review*, **88** 2, pp 135-170.

[7] Sutton, RS & Barto, AG (1987). A temporal-difference model of classical conditioning. *Proceedings of the Ninth Annual Conference of the Cognitive Science Society*. Seattle, WA.
